# Generalization Bounds and Consistency for Latent Structural Probit and Ramp Loss

**David McAllester**
TTI-Chicago
mcallester@ttic.edu

**Joseph Keshet**
TTI-Chicago
jkeshet@ttic.edu

## Abstract

We consider latent structural versions of probit loss and ramp loss. We show that these surrogate loss functions are consistent in the strong sense that for any feature map (finite or infinite dimensional) they yield predictors approaching the infimum task loss achievable by any linear predictor over the given features. We also give finite sample generalization bounds (convergence rates) for these loss functions. These bounds suggest that probit loss converges more rapidly. However, ramp loss is more easily optimized on a given sample.

## 1 Introduction

Machine learning has become a central tool in areas such as speech recognition, natural language translation, machine question answering, and visual object detection. In modern approaches to these applications systems are evaluated with quantitative performance metrics. In speech recognition one typically measures performance by the word error rate. In machine translation one typically uses the BLEU score. Recently the IBM deep question answering system was trained to optimize the Jeopardy game show score. The PASCAL visual object detection challenge is scored by average precision in recovering object bounding boxes. No metric is perfect and any metric is controversial, but quantitative metrics provide a basis for quantitative experimentation and quantitative experimentation has lead to real progress. Here we adopt the convention that a performance metric is given as a *task loss* — a measure of a quantity of error or cost such as the word error rate in speech recognition. We consider general methods for minimizing task loss at evaluation time.

Although the goal is to minimize task loss, most systems are trained by minimizing a *surrogate loss* different from task loss. A surrogate loss is necessary when using scale-sensitive regularization in training a linear classifier. A linear classifier selects the output that maximizes an inner product of a feature vector and a weight vector. The output of a linear classifier does not change when the weight vector is scaled down. But for most regularizers of interest, such as a norm of the weight vector, scaling down the weight vector drives the regularizer to zero. So directly regularizing the task loss of a linear classifier is meaningless.

For binary classification standard surrogate loss functions include log loss, hinge loss, probit loss, and ramp loss. Unlike binary classification, however, the applications mentioned above involve complex (or structured) outputs. The standard surrogate loss functions for binary classification have generalizations to the structured output setting. Structural log loss is used in conditional random fields (CRFs) [7]. Structural hinge loss is used in structural SVMs [13, 14]. Structural probit loss is defined and empirically evaluated in [6]. A version of structural ramp loss is defined and empirically evaluated in [3] (but see also [12] for a treatment of the fundamental motivation for ramp loss). All four of these structural surrogate loss functions are defined formally in section 2.[1]

This paper is concerned with developing a better theoretical understanding of the relationship between surrogate loss training and task loss testing for structured labels. Structural ramp loss is justified in [3] as being a tight upper bound on task loss. But of course the tightest upper bound on task loss is the task loss itself. Here we focus on generalization bounds and consistency. A finite sample generalization bound for probit loss was stated implicitly in [9] and an explicit probit loss bound is given in [6]. Here we review the finite sample bounds for probit loss and prove a finite sample bound for ramp loss. Using these bounds we show that probit loss and ramp loss are both consistent in the sense that for any arbitrary feature map (possibly infinite dimensional) optimizing these surrogate loss functions with appropriately weighted regularization approaches, in the limit of infinite training data, the minimum loss achievable by a linear predictor over the given features. No convex surrogate loss function, such as log loss or hinge loss, can be consistent in this sense — for any nontrivial convex surrogate loss function one can give examples (a single feature suffices) where the learned weight vector is perturbed by outliers but where the outliers do not actually influence the optimal task loss.

Both probit loss and ramp loss can be optimized in practice by stochastic gradient descent. Ramp loss is simpler and easier to implement. The subgradient update for ramp loss is similar to a perceptron update — the update is a difference between a "good" feature vector and a "bad" feature vector. Ramp loss updates are closely related to updates derived from n-best lists in training machine translaiton systems [8, 2]. Ramp loss updates regularized by early stopping have been shown to be effective in phoneme alignment [10]. It is also shown in [10] that in the limit of large weight vectors the expected ramp loss update converges to the true gradient of task loss. This result suggests consistency for ramp loss, a suggestion confirmed here. A practical stochastic gradient descent algorithm for structural probit loss is given in [6] where it is also shown that probit loss can be effective for phoneme recognition. Although the generalization bounds suggest that probit loss converges faster than ramp loss, ramp loss seems easier to optimize.

We formulate all the notions of loss in the presence of latent structure as well as structured labels. Latent structure is information that is not given in the labeled data but is constructed by the prediction algorithm. For example, in natural language translation the alignment between the words in the source and the words in the target is not explicitly given in a translation pair. Grammatical structures are also not given in a translation pair but may be constructed as part of the translation process. In visual object detection the position of object parts is not typically annotated in the labeled data but part position estimates may be used as part of the recognition algorithm. Although the presence of latent structure makes log loss and hinge loss non-convex, latent strucure seems essential in many applications. Latent structural log loss, and the notion of a hidden CRF, is formulated in [11]. Latent structural hinge loss, and the notion of a latent structural SVM, is formulated in [15].

## 2 Formal Setting and Review

We consider an arbitrary input space $\mathcal{X}$ and a finite label space $\mathcal{Y}$. We assume a source probability distribution over labeled data, i.e., a distribution over pairs $(x, y)$, where we write $\mathbb{E}_{x,y}\left[f(x, y)\right]$ for the expectation of $f(x, y)$. We assume a loss function $L$ such that for any two labels $y$ and $\hat{y}$ we have that $L(y, \hat{y}) \in [0, 1]$ is the loss (or cost) when the true label is $y$ and we predict $\hat{y}$. We will work with infinite dimensional feature vectors. We let $\ell_2$ be the set of finite-norm infinite-dimensional vectors — the set of all square-summable infinite sequences of real numbers. We will be interested in linear predictors involving latent structure. We assume a finite set $\mathcal{Z}$ of "latent labels". For example, we might take $\mathcal{Z}$ to be the set of all parse trees of source and target sentences in a machien translation system. In machine translation the label $y$ is typically a sentence with no parse tree specified. We can recover the pure structural case, with no latent information, by taking $\mathcal{Z}$ to be a singleton set. It will be convenient to define $\mathcal{S}$ to be the set of pairs of a label and a latent label. An element $s$ of $\mathcal{S}$ will be called an *augmented label* and we define $L(y, s)$ by $L(y, (\hat{y}, z)) = L(y, \hat{y})$. We assume a feature map $\phi$ such that for an input $x$ and augmented label $s$ we have $\phi(x, s) \in \ell_2$ with $||\phi(x, s)|| \leq 1$.[2]

Given an input $x$ and a weight vector $w \in \ell_2$ we define the prediction $\hat{s}_w(x)$ as follows.

$$\hat{s}_w(x) \quad = \quad \operatorname*{argmax}_s w^\top \phi(x, s)$$

Our goal is to use the training data to learn a weight vector $w$ so as to minimize the expected loss on newly drawn labeled data $\mathbb{E}_{x,y}\left[L(y, \hat{s}_w(x))\right]$. We will assume an infinite sequence of training data $(x_1, y_1)$, $(x_2, y_2)$, $(x_3, y_3)$, ... drawn IID from the source distribution and use the following notations.

$$L(w, x, y) = L(y, \hat{s}_w(x)) \qquad L(w) = \mathbb{E}_{x,y}\left[L(w, x, y)\right]$$

$$L^* = \inf_{w \in \ell_2} L(w) \qquad \hat{L}^n(w) = \frac{1}{n}\sum_{i=1}^n L(w, x_i, y_i)$$

We adopt the convention that in the definition of $L(w, x, y)$ we break ties in definition of $\hat{s}_w(x)$ in favor of augmented labels of larger loss. We will refer to this as *pessimistic tie breaking*.

Here we define latent structural log loss, hinge loss, ramp loss and probit loss as follows.

$$
\begin{aligned}
L_{\log}(w, x, y) &= \ln \frac{1}{P_w(y|x)} = \ln Z_w(x) - \ln Z_w(x, y) \\
& \quad Z_w(x) = \sum_s \exp(w^\top \Phi(x, s)) \quad Z_w(x, y) = \sum_z \exp(w^\top \phi(x, (y, z))) \\
L_{\text{hinge}}(w, x, y) &= \left(\max_s w^\top \phi(x, s) + L(y, s)\right) - \left(\max_z w^\top \Phi(x, (y, z))\right) \\
L_{\text{ramp}}(w, x, y) &= \left(\max_s w^\top \phi(x, s) + L(y, s)\right) - \left(\max_s w^\top \Phi(x, s)\right) \\
&= \left(\max_s w^\top \phi(x, s) + L(y, s)\right) - w^\top \Phi(x, \hat{s}_w(x)) \\
L_{\text{probit}}(w, x, y) &= \mathbb{E}_\epsilon\left[L(y, \hat{s}_{w+\epsilon}(x))\right]
\end{aligned}
$$

In the definition of probit loss we take $\epsilon$ to be zero-mean unit-variance isotropic Gaussian noise — for each feature dimension $j$ we have that $\epsilon_j$ an independent zero-mean unit-variance Gaussian variable.[3] More generally we will write $\mathbb{E}_\epsilon\left[f(\epsilon)\right]$ for the expectation of $f(\epsilon)$ where $\epsilon$ is Gaussian noise. It is interesting to note that $L_{\log}$, $L_{\text{hinge}}$, and $L_{\text{ramp}}$ are all naturally differences of convex functions and hence can be optimized by CCCP.

In the case of binary classification we have $\mathcal{S} = \mathcal{Y} = \{-1, 1\}$, $\phi(x, y) = \frac{1}{2}y\phi(x)$, $L(y, y') = 1_{y \neq y'}$ and we define the margin $m = yw^\top \phi(x)$. We then have the following where the expression for $L_{\text{probit}}(w, x, y)$ assumes $||\Phi(x)|| = 1$.

$$L_{\log}(w, x, y) = \ln\left(1 + e^{-m}\right) \qquad L_{\text{hinge}}(w, x, y) = \max(0, 1 - m)$$

$$L_{\text{ramp}}(w, x, y) = \min(1, \max(0, 1 - m)) \quad L_{\text{probit}}(w, x, y) = P_{\epsilon \sim \mathcal{N}(0,1)}[\epsilon \geq m]$$

Returning to the general case we consider the relationship between hinge and ramp loss. First we consider the case where $\mathcal{Z}$ is a singleton set — the case of no latent structure. In this case hinge loss is convex in $w$ — the hinge loss becomes a maximum of linear functions. Ramp loss, however, remains a difference of nonlinear convex functions even for $\mathcal{Z}$ singleton. Also, in the case where $Z$ is singleton one can easily see that hinge loss is unbounded — wrong labels may score arbitrarily better than the given label. Hinge loss remains unbounded in case of non-singleton $\mathcal{Z}$. Ramp loss, on the other hand, is bounded by 1 as follows.

$$
\begin{aligned}
L_{\text{ramp}}(w, x, y) &= \left(\max_s w^\top \Phi(x, s) + L(y, s)\right) - w^\top \Phi(x, \hat{s}_w(x)) \\
&\leq \left(\max_s w^\top \Phi(x, s) + 1\right) - w^\top \Phi(x, \hat{s}_w(x)) = 1
\end{aligned}
$$

Next, as is emphasized in [3], we note that ramp loss is a tighter upper bound on task loss than is hinge loss. To see this we first note that it is immediate that $L_{\text{hinge}}(w, x, y) \geq L_{\text{ramp}}(w, x, y)$.

Furthermore, the following derivation shows $L_{\mathrm{ramp}}(w, x, y) \geq L(w, x, y)$ where we assume pessimistic tie breaking in the definition of $\hat{s}_w(x)$.

$$L_{\mathrm{ramp}}(w, x, y) = \left( \max_s w^\top \Phi(x, s) + L(y, s) \right) - w^\top \Phi(x, \hat{s}_w(x))$$

$$\geq w^\top \Phi(x, \hat{s}_w(x)) + L(y, \hat{s}_w(x)) - w^\top \Phi(x, \hat{s}_w(x)) = L(y, \hat{s}_w(x))$$

But perhaps the most important property of ramp loss is the following.

$$\lim_{\alpha \to \infty} L_{\mathrm{ramp}}(\alpha w, x, y) = L(w, x, y) \tag{1}$$

This can be verified by noting that as $\alpha$ goes to infinity the maximum of the first term in ramp loss must occur at $s = \hat{s}_w(x)$.

Next we note that Optimizing $L_{\mathrm{ramp}}$ through subgradient descent (rather than CCCP) yields the following update rule (here we ignore regularization).

$$\Delta w \propto \phi(x, \hat{s}_w(x)) - \phi(x, \hat{s}_w^+(x, y)) \tag{2}$$
$$\hat{s}_w^+(x, y) = \underset{s}{\mathrm{argmax}}\, w^\top \phi(x, s) + L(y, s)$$

We will refer to (2) as the ramp loss update rule. The following is proved in [10] under mild conditions on the probability distribution over pairs $(x, y)$.

$$\nabla_w L(w) = \lim_{\alpha \to \infty} \alpha \mathbb{E}_{x,y} \left[ \phi(x, \hat{s}_{\alpha w}^+(x, y)) - \phi(x, \hat{s}_w(x)) \right] \tag{3}$$

Equation (3) expresses a relationship between the expected ramp loss update and the gradient of generalization loss. Significant empirical success has been achieved with the ramp loss update rule using early stopping regularization [10]. But both (1) and (3) suggests that regularized ramp loss should be consistent as is confirmed here.

Finally it is worth noting that $L_{\mathrm{ramp}}$ and $L_{\mathrm{probit}}$ are meaningful for an arbitrary prediction space $\mathcal{S}$, label space $\mathcal{Y}$, and loss function $L(y, s)$ between a label and a prediction. Log loss and hinge loss can be generalized to arbitrary prediction and label spaces provided that we assume a compatibility relation between predictions and labels. The framework of independent prediction and label spaces is explored more fully in [5] where a notion of weak-label SVM is defined subsuming both ramp and hinge loss as special cases.

## 3 Consistency of Probit Loss

We start with the consistency of probit loss which is easier to prove. We consider the following learning rule where the regularization parameter $\lambda_n$ is some given function of $n$.

$$\hat{w}_n = \underset{w}{\mathrm{argmin}}\ \ \hat{L}_{\mathrm{probit}}^n(w)\ \ +\ \frac{\lambda_n}{2n} ||w||^2 \tag{4}$$

We now prove the following fairly straightforward consequence of a generalization bound appearing in [6].

**Theorem 1** (Consistency of Probit loss). *For $\hat{w}_n$ defined by (4), if the sequence $\lambda_n$ increases without bound, and $\lambda_n \ln n / n$ converges to zero, then with probability one over the draw of the infinite sample we have $\lim_{n \to \infty} L_{\mathrm{probit}}(\hat{w}_n) = L^*$.*

Unfortunately, and in contrast to simple binary SVMs, for a latent binary SVM (an LSVM) there exists an infinite sequence $w_1, w_2, w_3, \dots$ such that $L_{\mathrm{probit}}(w_n)$ approaches $L^*$ but $L(w_n)$ remains bounded away from $L^*$ (we omit the example here). However, the learning algorithm (4) achieves consistency in the sense that the stochastic predictor defined by $\hat{w}_n + \epsilon$ where $\epsilon$ is Gaussian noise has a loss which converges to $L^*$.

To prove theorem 1 we start by reviewing the generalization bound of [6]. The departure point for this generalization bound is the following PAC-Bayesian theorem where $P$ and $Q$ range over probability measures on a given space of predictors and $L(Q)$ and $\hat{L}^n(Q)$ are defined as expectations over selecting a predictor from $Q$.

**Theorem 2** (from [1], see also [4]). *For any fixed prior distribution $P$ and fixed $\lambda > 1/2$ we have that with probability at least $1 - \delta$ over the draw of the training data the following holds simultaneously for all $Q$.*

$$L(Q) \leq \frac{1}{1 - \frac{1}{2\lambda}} \left( \hat{L}^n(Q) + \lambda \left( \frac{KL(Q, P) + \ln \frac{1}{\delta}}{n} \right) \right) \tag{5}$$

For the space of linear predictors we take the prior $P$ to be the zero-mean unit-variance Gaussian distribution and for $w \in \ell_2$ we define the distribution $Q_w$ to be the unit-variance Gaussian centered at $w$. This gives the following corollary of (5).

**Corollary 1** (from [6]). *For fixed $\lambda_n > 1/2$ we have that with probability at least $1 - \delta$ over the draw of the training data the following holds simultaneously for all $w \in \ell_2$.*

$$L_{\text{probit}}(w) \leq \frac{1}{1 - \frac{1}{2\lambda_n}} \left( \hat{L}^n_{\text{probit}}(w) + \lambda_n \left( \frac{\frac{1}{2}||w||^2 + \ln \frac{1}{\delta}}{n} \right) \right) \tag{6}$$

To prove theorem 1 from (6) we consider an arbitrary unit-norm weight vector $w^*$ and an arbitrary scalar $\alpha > 0$. Setting $\delta$ to $1/n^2$, and noting that $\hat{w}_n$ is the minimizer of the right hand side of (6), we have the following with probability at least $1 - 1/n^2$ over the draw of the sample.

$$L_{\text{probit}}(\hat{w}_n) \leq \frac{1}{1 - \frac{1}{2\lambda_n}} \left( \hat{L}^n_{\text{probit}}(\alpha w^*) + \lambda_n \left( \frac{\frac{1}{2}\alpha^2 + 2\ln n}{n} \right) \right) \tag{7}$$

A standard Chernoff bound argument yields that for $w^*$ and $\alpha > 0$ selected prior to drawing the sample, we have the following with probability at least $1 - 1/n^2$ over the choice of the sample.

$$\hat{L}^n_{\text{probit}}(\alpha w^*) \leq L_{\text{probit}}(\alpha w^*) + \sqrt{\frac{\ln n}{n}} \tag{8}$$

Combining (7) and (8) with a union bound yields that with probability at least $1 - 2/n^2$ we have the following.

$$L_{\text{probit}}(\hat{w}_n) \leq \frac{1}{1 - \frac{1}{2\lambda_n}} \left( L_{\text{probit}}(\alpha w^*) + \sqrt{\frac{\ln n}{n}} + \lambda_n \left( \frac{\frac{1}{2}\alpha^2 + 2\ln n}{n} \right) \right)$$

Because the probability that the above inequality is violated goes as $1/n^2$, with probability one over the draw of the sample we have the following.

$$\lim_{n \to \infty} L_{\text{probit}}(\hat{w}_n) \leq \lim_{n \to \infty} \frac{1}{1 - \frac{1}{2\lambda_n}} \left( L_{\text{probit}}(\alpha w^*) + \sqrt{\frac{\ln n}{n}} + \lambda_n \left( \frac{\frac{1}{2}\alpha^2 + 2\ln n}{n} \right) \right)$$

Under the conditions on $\lambda_n$ given in the statement of theorem 1 we then have

$$\lim_{n \to \infty} L_{\text{probit}}(\hat{w}_n) \leq L_{\text{probit}}(\alpha w^*).$$

Because this holds with probability one for any $\alpha$, the following must also hold with probability one.

$$\lim_{n \to \infty} L_{\text{probit}}(\hat{w}_n) \leq \lim_{\alpha \to \infty} L_{\text{probit}}(\alpha w^*) \tag{9}$$

Now consider

$$\lim_{\alpha \to \infty} L_{\text{probit}}(\alpha w, x, y) = \lim_{\alpha \to \infty} \mathbb{E}_\epsilon \left[ L(\alpha w + \epsilon, x, y) \right] = \lim_{\sigma \to 0} \mathbb{E}_\epsilon \left[ L(w + \sigma\epsilon, x, y) \right].$$

We have that $\lim_{\sigma \to 0} \mathbb{E}_\epsilon \left[ L(w + \sigma\epsilon, x, y) \right]$ is determined by the augmented labels $s$ that are tied for the maximum value of $w^\top \Phi(x, s)$. There is some probability distribution over these tied values that occurs in the limit of small $\sigma$. Under the pessimistic tie breaking in the definition of $L(w, x, y)$ we then get $\lim_{\alpha \to \infty} L_{\text{probit}}(\alpha w, x, y) \leq L(w, x, y)$. This in turn gives the following.

$$\lim_{\alpha \to \infty} L_{\text{probit}}(\alpha w) = \mathbb{E}_{x,y} \left[ \lim_{\alpha \to \infty} L_{\text{probit}}(\alpha w, x, y) \right] \leq \mathbb{E}_{x,y} \left[ L(w, x, y) \right] = L(w) \tag{10}$$

Combining (9) and (10) yields $\lim_{n \to \infty} L_{\text{probit}}(\hat{w}_n) \leq L(w^*)$. Since for any $w^*$ this holds with probability one, with probability one we also have $\lim_{n \to \infty} L_{\text{probit}}(\hat{w}_n) \leq L^*$. Finally we note $L_{\text{probit}}(w) = \mathbb{E}_\epsilon \left[ L(w + \epsilon) \right] \geq L^*$ which then gives theorem 1.

# 4 Consistency of Ramp Loss

Now we consider the following ramp loss training equation.

$$\hat{w}_n = \underset{w}{\operatorname{argmin}}\ \ \hat{L}_{\text{ramp}}^n(w)\ \ +\ \ \frac{\gamma_n}{2n}\,||w||^2 \tag{11}$$

The main result of this paper is the following.

**Theorem 3** (Consistency of Ramp Loss). *For $\hat{w}_n$ defined by (11), if the sequence $\gamma_n/\ln^2 n$ increases without bound, and the sequence $\gamma_n/(n\ln n)$ converges to zero, then with probability one over the draw of the infinite sample we have $\lim_{n\to\infty} L_{\text{probit}}((\ln n)\hat{w}_n) = L^*$.*

As with theorem 1, theorem 3 is derived from a finite sample generalization bound. The bound is derived from (6) by upper bounding $\hat{L}_{\text{probit}}^n(w/\sigma)$ in terms of $\hat{L}_{\text{ramp}}^n(w)$. From section 3 we have that $\lim_{\sigma\to 0} L_{\text{probit}}(w/\sigma, x, y) \leq L(w, x, y) \leq L_{\text{ramp}}(w, x, y)$. This can be converted to the following lemma for finite $\sigma$ where we recall that $\mathcal{S}$ is the set of augmented labels $s = (y, z)$.

**Lemma 1.**

$$L_{\text{probit}}\left(\frac{w}{\sigma}, x, y\right) \leq L_{\text{ramp}}(w, x, y) + \sigma + \sigma\sqrt{8\ln\frac{|\mathcal{S}|}{\sigma}}$$

*Proof.* We first prove that for any $\sigma > 0$ we have

$$L_{\text{probit}}\left(\frac{w}{\sigma}, x, y\right) \leq \sigma + \max_{s:\ m(s)\leq M} L(y, s) \tag{12}$$

where

$$m(s) = w^\top \Delta\phi(s) \quad \Delta\phi(s) = \phi(x,\ \hat{s}_w(x)) - \phi(x,\ s) \quad M = \sigma\sqrt{8\ln\frac{|\mathcal{S}|}{\sigma}}.$$

To prove (12) we note that for $m(s) > M$ we have the following where $P_\epsilon[\Phi(\epsilon)]$ abbreviates $\mathbb{E}_\epsilon\left[1_{\Phi(\epsilon)}\right]$.

$$
\begin{aligned}
P_\epsilon[\hat{s}_{w+\sigma\epsilon}(x) = s] &\leq& P_\epsilon[(w+\sigma\epsilon)^\top \Delta\phi(s) \leq 0] = P_\epsilon\left[-\epsilon^\top\Delta\phi(s) \geq m(s)/\sigma\right] \\
&\leq& P_{\epsilon\sim\mathcal{N}(0,1)}\left[\epsilon \geq \frac{M}{2\sigma}\right] \leq \exp\left(-\frac{M^2}{8\sigma^2}\right) = \frac{\sigma}{|\mathcal{S}|}
\end{aligned}
$$

$$
\begin{aligned}
\mathbb{E}_\epsilon\left[L(y, \hat{s}_{w+\sigma\epsilon}(x))\right] &\leq& P_\epsilon\left[\exists s:\ m(s) > M\ \ \hat{s}_{w+\epsilon\sigma}(x) = s\right] + \max_{s:m(s)\leq M} L(y, s) \\
&\leq& \sigma + \max_{s:m(s)\leq M} L(y, s)
\end{aligned}
$$

The following calculation shows that (12) implies the lemma.

$$
\begin{aligned}
L_{\text{probit}}\left(\frac{w}{\sigma}, x, y\right) &\leq& \sigma + \max_{s:\ m(s)\leq M} L(y, s) \\
&\leq& \sigma + \left(\max_{s:\ m(s)\leq M} L(y, s) - m(s)\right) + M \\
&\leq& \sigma + \left(\max_s L(y, s) - m(s)\right) + M \\
&=& \sigma + L_{\text{ramp}}(w, x, y) + M
\end{aligned}
$$

$\square$

Inserting lemma 1 into (6) we get the following.

**Theorem 4.** *For $\lambda_n > 1/2$ we have that with probability at least $1 - \delta$ over the draw of the training data the following holds simultaneously for all $w$ and $\sigma > 0$.*

$$L_{\text{probit}}\left(\frac{w}{\sigma}\right) \leq \frac{1}{1 - \frac{1}{2\lambda_n}}\left(\hat{L}_{\text{ramp}}^n(w) + \sigma + \sigma\sqrt{8\ln\frac{|\mathcal{S}|}{\sigma}} + \lambda_n\left(\frac{\frac{||w||^2}{2\sigma^2} + \ln\frac{1}{\delta}}{n}\right)\right) \tag{13}$$

To prove theorem 3 we now take $\sigma_n = 1/\ln n$ and $\lambda_n = \gamma_n/\ln^2 n$. We then have that $\hat{w}_n$ is the minimizer of the right hand side of (13). This observation yields the following for any unit-norm vector $w^*$ and scalar $\alpha > 0$ where we have set $\delta = 1/n^2$.

$$L_{\mathrm{probit}}((\ln n)\hat{w}_n) \leq \frac{1}{1 - \frac{\ln^2 n}{2\gamma_n}} \left( \hat{L}_{\mathrm{ramp}}(\alpha w^*) + \frac{1 + \sqrt{8\ln(|\mathcal{S}|\ln n)}}{\ln n} + \frac{\gamma_n \alpha^2}{2n} + \frac{2\gamma_n}{n \ln n} \right) \quad (14)$$

As in section 3, we use a Chernoff bound for the single vector $w^*$ and scalar $\alpha$ to bound $\hat{L}_{\mathrm{ramp}}(\alpha w^*)$ in terms of $L_{\mathrm{ramp}}(\alpha w^*)$ and then take the limit as $n \to \infty$ to get the following with probability one.

$$\lim_{n\to\infty} L_{\mathrm{probit}}((\ln n)\hat{w}_n) \leq L_{\mathrm{ramp}}(\alpha w^*)$$

The remainder of the proof is the same in section 3 but where we now use $\lim_{\alpha\to\infty} L_{\mathrm{ramp}}(\alpha w^*) = L(w^*)$ whose proof we omit.

## 5   A Comparison of Convergence Rates

To compare the convergence rates implicit in (6) and (13) we note that in (13) we can optimize $\sigma$ as a function of other quantities in the bound.[4] An approximately optimal value for $\sigma$ is $\left(\lambda_n ||w||^2/n\right)^{1/3}$ which gives the following.

$$L_{\mathrm{probit}}\left(\frac{w}{\sigma}\right) \leq \frac{1}{1 - \frac{1}{2\lambda_n}} \left( \hat{L}_{\mathrm{ramp}}^n(w) + \left(\frac{\lambda_n ||w||^2}{n}\right)^{1/3} \left( \frac{3}{2} + \sqrt{8\ln \frac{|\mathcal{S}|}{\sigma}} \right) + \frac{\lambda_n \ln \frac{1}{\delta}}{n} \right) \quad (15)$$

We have that (15) gives $\tilde{O}\left(\left(||\hat{w}_n||^2/n\right)^{1/3}\right)$ as opposed to (6) which gives $O\left(||\hat{w}_n||^2/n\right)$. This suggests that while probit loss and ramp loss are both consistent, ramp loss may converge more slowly.

## 6   Discussion and Open Problems

The contributions of this paper are a consistency theorem for latent structural probit loss and both a generalization bound and a consistency theorem for latent structural ramp loss. These bounds suggest that probit loss converges more rapidly. However, we have only proved upper bounds on generalization loss and it remains possible that these upper bounds, while sufficient to show consistency, are not accurate characterizations of the actual generalization loss. Finding more definitive statements, such as matching lower bounds, remains an open problem.

The definition of ramp loss used here is not the only one possible. In particular we can consider the following variant.

$$L'_{\mathrm{ramp}}(w,x,y) = \left( \max_s w^\top \Phi(x,s) \right) - \left( \max_s w^\top \phi(x,s) - L(y,s) \right)$$

Relations (1) and (3) both hold for $L'_{\mathrm{ramp}}$ as well as $L_{\mathrm{ramp}}$. Experiments indicate that $L'_{\mathrm{ramp}}$ performs somewhat better than $L_{\mathrm{ramp}}$ under early stopping of subgradient descent. However it seems that it is not possible to prove a bound of the form of (15) for $L'_{\mathrm{ramp}}$. A frustrating observation is that $L'_{\mathrm{ramp}}(0,x,y) = 0$. Finding a meaningful finite-sample statement for $L'_{\mathrm{ramp}}$ remains an open problem.

The isotropic Gaussian noise distribution used in the definition of $L_{\mathrm{probit}}$ is not optimal. A uniformly tighter upper bound on generalization loss is achieved by optimizing the posterior in the PAC-Bayesian theorem. Finding a practical more optimal use of the PAC-Bayesian theorem also remains an open problem.

## Footnotes

[1]The definition of ramp loss used here is slightly different from that in [3].

[2]We note that this setting covers the finite dimensional case because the range of the feature map can be taken to be a finite dimensional subset of $\ell_2$ — we are not assuming a universal feature map.

[3]In infinite dimension we have that with probability one $||\epsilon|| = \infty$ and hence $w + \epsilon$ is not in $\ell_2$. The measure underling $\mathbb{E}_\epsilon\left[f(\epsilon)\right]$ is a Gaussian process. However, we still have that for any unit-norm feature vector $\Phi$ the inner product $\epsilon^\top \Phi$ is distributed as a zero-mean unit-norm scalar Gaussian and $L_{\text{probit}}(w, x, y)$ is therefore well defined.

[4]In the consistency proof it was more convenient to set $\sigma = 1/ln\ n$ which is plausibly nearly optimal anyway.

# References

[1] Olivier Catoni. *PAC-Bayesian Supervised Classification: The Thermodynamics of Statistical Learning*. Institute of Mathematical Statistics LECTURE NOTES MONOGRAPH SERIES, 2007.

[2] D. Chiang, K. Knight, and W. Wang. 11,001 new features for statistical machine translation. In *Proc. NAACL, 2009*, 2009.

[3] Chuong B. Do, Quoc Le, Choon Hui Teo, Olivier Chapelle, and Alex Smola. Tighter bounds for structured estimation. In *nips*, 2008.

[4] Pascal Germain, Alexandre Lacasse, Francois Laviolette, and Mario Marchand. Pac-bayesian learning of linear classifiers. In *ICML*, 2009.

[5] Ross Girshick, Pedro Felzenszwalb, and David McAllester. Object detection with grammar models. In *NIPS*, 2011.

[6] Joseph Keshet, David McAllester, and Tamir Hazan. Pac-bayesian approach for minimization of phoneme error rate. In *International Conference on Acoustics, Speech, and Signal Processing (ICASSP)*, 2011.

[7] J. Lafferty, A. McCallum, and F. Pereira. Conditional random fields: Probabilistic models for segmenting and labeling sequence data. In *Proceedings of the Eighteenth International Conference on Machine Learning*, pages 282–289, 2001.

[8] P. Liang, A. Bouchard-Ct, D. Klein, and B. Taskar. An end-to-end discriminative approach to machine translation. In *International Conference on Computational Linguistics and Association for Computational Linguistics (COLING/ACL)*, 2006.

[9] David McAllester. Generalization bounds and consistency for structured labeling. In G. Bakir nd T. Hofmann, B. Scholkopf, A. Smola, B. Taskar, and S. V. N. Vishwanathan, editors, *Predicting Structured Data*. MIT Press, 2007.

[10] David A. McAllester, Tamir Hazan, and Joseph Keshet. Direct loss minimization for structured prediction. In *Advances in Neural Information Processing Systems 24*, 2010.

[11] A. Quattoni, S. Wang, L.P. Morency, M Collins, and T Darrell. Hidden conditional random fields. *PAMI*, 29, 2007.

[12] R.Collobert, F.H.Sinz, J.Weston, and L.Bottou. Trading convexity for scalability. In *ICML*, 2006.

[13] B. Taskar, C. Guestrin, and D. Koller. Max-margin markov networks. In *Advances in Neural Information Processing Systems 17*, 2003.

[14] I. Tsochantaridis, T. Hofmann, T. Joachims, and Y. Altun. Support vector machine learning for interdependent and structured output spaces. In *Proceedings of the Twenty-First International Conference on Machine Learning*, 2004.

[15] Chun-Nam John Yu and T. Joachims. Learning structural svms with latent variables. In *International Conference on Machine Learning (ICML)*, 2009.

